# Pulsestream Synapses with Non-Volatile Analogue Amorphous-Silicon Memories.

**A.J. Holmes, A.F. Murray, S. Churcher and J. Hajto**
Department of Electrical Engineering
University of Edinburgh
Edinburgh, EH9 3JL

**M. J. Rose**
Dept. of Applied Physics and Electronics,
Dundee University
Dundee DD1 4HN

## Abstract

A novel two-terminal device, consisting of a thin 1000Å layer of $p^+$ a-Si:H sandwiched between Vanadium and Chromium electrodes, exhibits a non-volatile, analogue memory action. This device stores synaptic weights in an ANN chip, replacing the capacitor previously used for dynamic weight storage. Two different synapse designs are discussed and results are presented.

## 1   INTRODUCTION

Analogue hardware implementations of neural networks have hitherto been hampered by the lack of a straightforward (local) analogue memory capability. The ideal storage mechanism would be compact, non-volatile, easily reprogrammable, and would not interfere with the normal silicon chip fabrication process.

Techniques which have been used to date include resistors (these are not generally reprogrammable, and suffer from being large and difficult to fabricate with any accuracy), dynamic capacitive storage [4] (this is compact, reprogrammable and simple, but implies an increase in system complexity, arising from off-chip refresh circuitry),

EEPROM ("floating gate") memory [5] (which is compact, reprogrammable, and non-volatile, but is slow, and cannot be reprogrammed in situ), and local digital storage (which is non-volatile, easily programmable and simple, but consumes area horribly).

Amorphous silicon has been used for synaptic weight storage [1, 2], but only as either a high-resistance fixed weight medium or a binary memory.

In this paper, we demonstrate that novel amorphous silicon memory devices can be incorporated into standard CMOS synapse circuits, to provide an analogue weight storage mechanism which is compact, non-volatile, easily reprogrammable, and simple to implement.

## 2    a-Si:H MEMORY DEVICES

The a-Si:H analogue memory device [3] comprises a 1000Å thick layer of amorphous silicon ($p^+$ a-Si:H) sandwiched between Vanadium and Chromium electrodes.

The a-Si device takes the form of a two-terminal, programmable resistor. It is an "add-on" to a conventional CMOS process, and does not demand that the normal CMOS fabrication cycle be disrupted. The a-Si device sits on top of the completed chip circuitry, making contact with the CMOS arithmetic elements via holes cut in the protective passivation layer, as shown in Figure 1.

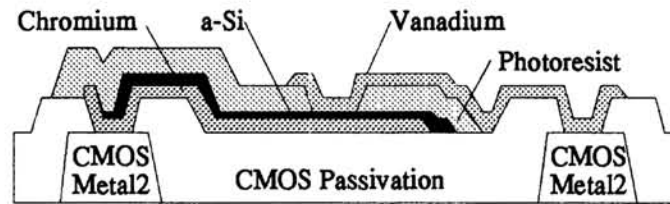

Figure 1: The construction of a-Si:H Devices on a CMOS chip

After fabrication a number of electronic procedures must be performed in order to program the device to a given resistance state.

**Programming, and Pre-Programming Procedures**

Before the a-Si device is usable, the following steps must be carried out:

- Forming: This is a once-only process, applied to the a-Si device in its "virgin" state, where it has a resistance of several MΩ. A series of 300ns pulses, increasing in amplitude from 5v to 14v, is applied to the device electrodes. This creates a vertical conducting channel or filament whose approximate resistance is 1KΩ. This filament can then be programmed to a value in the range 1KΩ to 1MΩ. The details of the physical mechanisms are not yet fully established, but it is clear that conduction occurs through a narrow (sub-micron) conducting channel.

- Write: To decrease the device's resistance, negative "Write", pulses are applied.

- Erase: To increase the device's resistance, positive "Erase", pulses are applied.

- Usage: Pulses below 0.5v do not change the device resistance. The resistance can therefore be utilised as a weight storage medium using a voltage of less than 0.5v without causing reprogramming.

Programming pulses, which range between 2v and 5v, are typically 120ns in duration. Programming is therefore much faster than for other EEPROM (floating gate) devices used in the same context, which use a series of $100\mu s$ pulses to set the threshold voltage [5].

The following sections describe synapse circuits using the a-Si:H devices. These synapses use the reprogrammable a-Si:H resistor in the place of a storage capacitor or EEPROM cell. These new synapses were implemented on a chip referred to as ASiTEST2, consisting of five main test blocks, each comprising of four synapses connected to a single neuron.

## 3    The EPSILON based synapse

The first synapse to be designed used the a-Si:H resistor as a direct replacement for the storage capacitor used in the EPSILON [4] synapse.

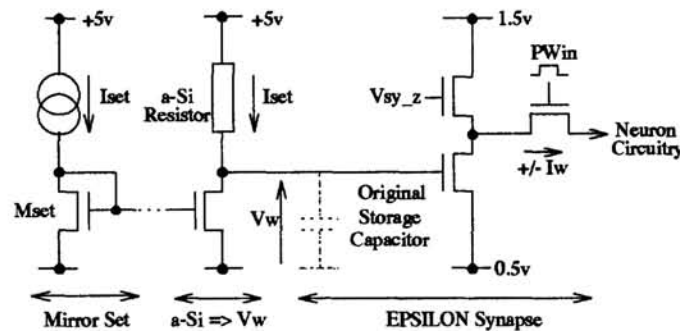

Figure 2: The EPSILON Synapse with a-Si:H weight storage

In the original EPSILON chip the weight voltage was stored as a voltage on a capacitor. In this new synapse design, shown in Figure 2, the a-Si:H resistance is set such that the voltage drop produced by Iset is equivalent to the original weight voltage, Vw, that was stored dynamically on the capacitor.

A new, simpler, synapse, which can be operated from a single +5v supply, was also be included on the ASiTEST2 chip.

## 4    The MkII synapse

The circuit is shown in Figure 3. The a-Si:H memory is used to store a current, Iasi. This current is subtracted from a zero current, Isy_z, to give a weight current , +/-Iw, which adds or subtracts charge from the activity capacitor, Cact, thus implementing excitation or inhibition respectively.

For the circuit to function correctly we must limit the voltage on the activity capacitor to the range [1.5v,3.5v], to ensure that the transistors mirroring Isy_z and Iasi remain in saturation. As Figure 3 shows, there are few reference signals and the circuit operates from a single +5v power supply rail, in sharp contrast to many earlier analogue neural circuits, including our own.

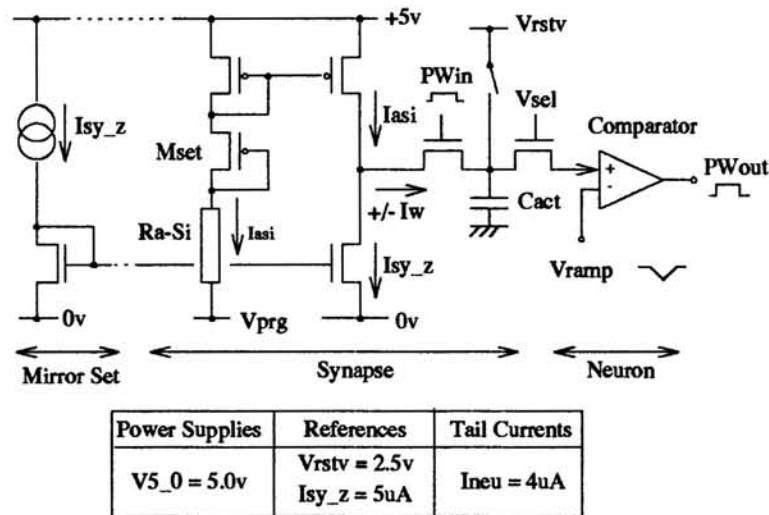

| Power Supplies | References | Tail Currents |
|---|---|---|
| V5_0 = 5.0v | Vrstv = 2.5v<br>Isy_z = 5uA | Ineu = 4uA |

Figure 3: The MkII synapse

On first inspection the main drawback of this design would appear to be a reliance on the accuracy with which the zero current Isy_z is mirrored across an entire chip. The variation in this current means that two cells with the same synapse resistance could produce widely differing values of Iw. However, during programming we do not use the resistance of the a-Si:H device as a target value. We monitor the **voltage on Cact** for a given PWin signal, increasing or decreasing the resistance of the a-Si:H device until the desired **voltage level** is achieved.

Example: To set a weight to be the maximum positive value, we adjust the a-Si resistance until a PWin signal of 5us, the maximum input signal, gives a voltage of 3.5v on the integration capacitor.

We are able to set the synapse weight using the whole integration range of [1.5v,3.5v] by only closing Vsel for the desired synapse during programming. In normal operating mode all four Vsel switches will be closed so that the integration charge is summed over all four local capacitors.

### 4.1   Example - Stability Test

As an example of the use of integration voltage as means of monitoring the resistance of a particular synapse we have included a stability test. This was carried out on one of the test chips which contained the MkII synapse.

The four synapses on the test chip were programmed to give different levels of activation. The chip was then powered up for 30mins each day during a 7-day period, and the activation levels for each synapse were measured three times.

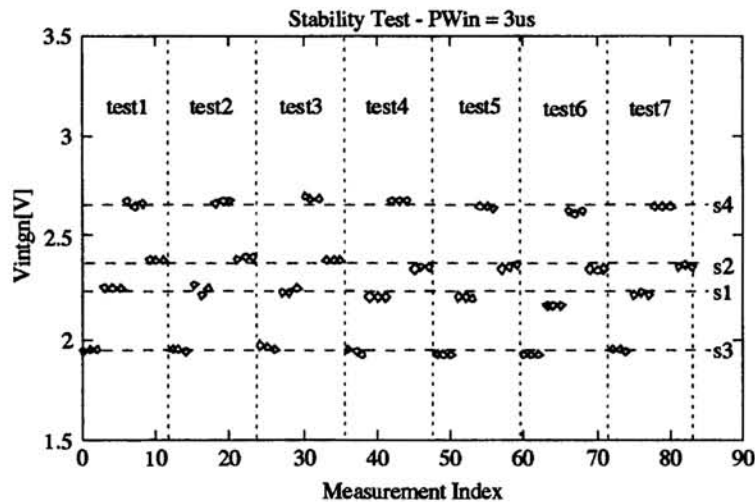

Figure 4: ASiTEST2- Stability Test

As figure 4 shows, the memories remain in the same resistance state (i.e retain their programmed weight value) over the whole 7-day period. Separate experiments on isolated devices indicate much longer hold times - of the order of months at least.

## 5   ASiTEST3

Recently we have received our latest, overtly neural, a-Si:H based test chip. This contains an 8x8 array of the MkII synapses.

The circuit board for this device has been constructed and partially tested while the ASiTEST3 chips are awaiting the deposition of the a-Si:H layers. We have been able to use an ASiTEST2 chip containing two of the MkII synapse test blocks i.e. 8 synapses and 2 neurons to exercise much of the board's functionality.

The test board contains a simple state machine which has four different states:

- State 0: Load Input Pulsewidths into SRAM from PC.
- State 1: Apply Input Pulsewidth signals to chip1.
- State 2: Use Vramp to generate threshold function for chip1. The resulting Pulsewidth outputs are used as the inputs to chip2, as well as being stored

in SRAM.

- State 3: Use Vramp to generate threshold function for chip2. Read resulting Pulsewidth Outputs into SRAM.
- State 0: Read Output Pulsewidths from SRAM into PC.

The results obtained during a typical test cycle are shown in Figure 5.

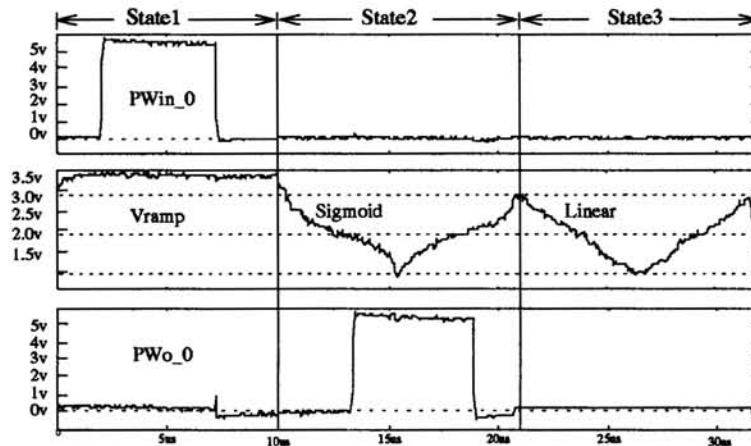

Figure 5: ASiTEST3 Board Scope Waveforms

As this figure shows different ramp signals, corresponding to different threshold functions, can be applied to chip1 and chip2 neurons.

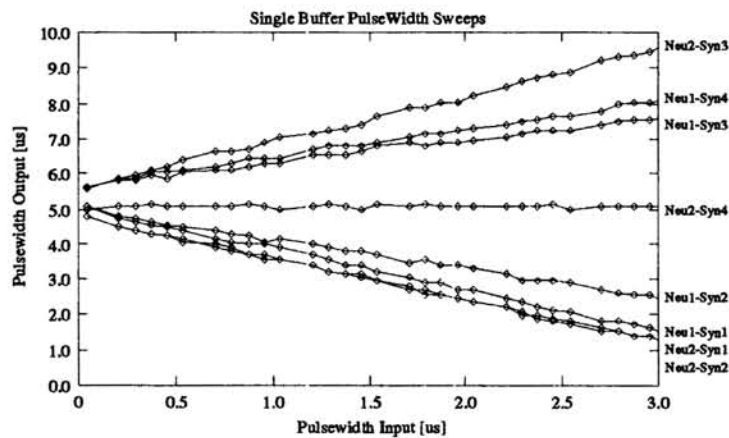

Figure 6: ASiTEST3 Board - MkII Synapse Characteristic

While the signals shown in Figure 5 appear noisy the multiplier characteristic that the chip produces is still admirably linear, as shown in Figure 6. In this experiment all eight synapses on a test chip were programmed into different resistance states and PWin was swept from 0 to 3us.

# 6  Conclusions

We have demonstrated the use of novel a-Si:H analogue memory devices as a means of storing synaptic weights in a Pulsewidth ANN. We have also demonstrated the operation of an interface board which allows two 8x8 ANN chips, operating as a two layer network, to be controlled by a simple PC interface card.

This technology is most suitable for small networks in, for example, remote control and other embedded-system applications where cost and power considerations favour a single all-inclusive ANN chip with non-volatile, but programmable weights.

Another possible application of this technology is in large networks constructed using Thin Film Technology(TFT). If TFT's were used in place of the CMOS transistors then the area constraint imposed by crystalline silicon would be removed, allowing truly massively parallel networks to be integrated.

In summary - the a-Si:H analogue memory devices described in this paper provide a route to an analogue, non-volatile and fast synaptic weight storage medium. At the present time neither the programming nor storage mechanisms are fully understood making it difficult to compare this new device with more established technologies such as the ubiquitous Floating-Gate EEPROM technique. Current research is focused on firstly, improving the yield on the a-Si:H device which is unacceptably low at present, a demerit that we attribute to imperfections in the a-Si fabrication process and secondly, improving understanding of the device physics and hence the programming and storage mechanisms.

### Acknowledgements

This research has been jointly funded by BT, and EPSRC (formerly SERC), the Engineering and Physical Sciences Research Council.

# References

[1] W. Hubbard et al.(1986) Electronic Neural Networks *AIP Conference Proceedings - Snowbird 1986* :227-234

[2] H.P. Graf (1986) VLSI Implementation of a NN memory with several hundreds of neurons *AIP Conference Proceedings - Snowbird 1986* :182-187.

[3] M.J. Rose et al (1989) Amorphous Silicon Analogue Memory Devices *Journal of Non-Crystalline Solids* **1**(115):168-170

[4] A.Hamilton et al. (1992) Integrated Pulse-Stream Neural Networks - Results, Issues and Pointers *IEEE Transactions on N.N.s* **3**(3):385-393

[5] M.Holler, S.Tam, H.Castro and R.Benson (1989) An Electrically Trainable ANN with 10240 Floating Gate Synapses. *Int Conf on N.N.s Proc* :191-196

[6] A.F.Murray and A.V.W.Smith.(1987) Asynchronous Arithmetic for VLSI Neural Systems. *Electronics Letters* **23**(12):642-643

[7] A.J. Holmes et al. (1993) Use of a-Si:H Memory Devices for Non-volatile Weight Storage in ANNs. *Proc ICAS 15* :817-820

